# Investment Learning
# with Hierarchical PSOMs

**Jörg Walter** and **Helge Ritter**
Department of Information Science
University of Bielefeld, D-33615 Bielefeld, Germany
Email: {walter,helge}@techfak.uni-bielefeld.de

## Abstract

We propose a hierarchical scheme for rapid learning of context dependent "skills" that is based on the recently introduced *"Parameterized Self-Organizing Map"* ("PSOM"). The underlying idea is to first invest some learning effort to *specialize the system into a rapid learner* for a more restricted range of contexts.

The specialization is carried out by a prior "investment learning stage", during which the system acquires a set of basis mappings or "skills" for a set of prototypical contexts. Adaptation of a "skill" to a new context can then be achieved by interpolating in the space of the basis mappings and thus can be extremely rapid.

We demonstrate the potential of this approach for the task of a 3D visuo-motor map for a Puma robot and two cameras. This includes the forward and backward robot kinematics in 3D end effector coordinates, the 2D+2D retina coordinates and also the 6D joint angles. After the *investment phase* the transformation can be learned for a new camera set-up with a *single* observation.

## 1 Introduction

Most current applications of neural network learning algorithms suffer from a large number of required training examples. This may not be a problem when data are abundant, but in many application domains, for example in robotics, training examples are costly and the benefits of learning can only be exploited when significant progress can be made within a very small number of learning examples.

In the present contribution, we propose in section 3 a hierarchically structured learning approach which can be applied to many learning tasks that require system identification from a limited set of observations. The idea builds on the recently introduced *"Parameterized Self-Organizing Maps"* ("PSOMs"), whose strength is learning maps from a very small number of training examples [8, 10, 11].

In [8], the feasibility of the approach was demonstrated in the domain of robotics, among them, the learning of the inverse kinematics transform of a full 6-degree of freedom (DOF) Puma robot. In [10], two improvements were introduced, both achieve a significant increase in mapping accuracy and computational efficiency. In the next section, we give a short summary of the PSOM algorithm; it is decribed in more detail in [11] which also presents applications in the domain of visual learning.

## 2 The PSOM Algorithm

A *Parameterized Self-Organizing Map* is a parametrized, $m$-dimensional hyper-surface $M = \{\mathbf{w}(\mathbf{s}) \in X \subseteq \mathbb{R}^d | \mathbf{s} \in S \subseteq \mathbb{R}^m\}$ that is embedded in some higher-dimensional vector space $X$. $M$ is used in a very similar way as the standard discrete self-organizing map: given a distance measure $dist(\mathbf{x}, \mathbf{x}')$ and an input vector $\mathbf{x}$, a best-match location $\mathbf{s}^*(\mathbf{x})$ is determined by minimizing

$$\mathbf{s}^* := \operatorname*{argmin}_{\mathbf{s} \in S} \; dist(\mathbf{x}, \mathbf{w}(s)) \tag{1}$$

The associated "best-match vector" $\mathbf{w}(\mathbf{s}^*)$ provides the best approximation of input $\mathbf{x}$ in the manifold $M$. If we require $dist(\cdot)$ to vary only in a subspace $X^{in}$ of $X$ (i.e., $dist(\mathbf{x}, \mathbf{x}') = dist(\mathbf{P}\mathbf{x}, \mathbf{P}\mathbf{x}')$, where the diagonal matrix $\mathbf{P}$ projects into $X^{in}$), $\mathbf{s}^*(\mathbf{x})$ actually will only depend on $\mathbf{P}\mathbf{x}$. The projection $(1-\mathbf{P})\mathbf{w}(\mathbf{s}^*(\mathbf{x})) \in X^{out}$ of $\mathbf{w}(s^*(\mathbf{x}))$ lies in the orthogonal subspace $X^{out}$ can be viewed as a (non-linear) *associative completion of a fragmentary input* $\mathbf{x}$ of which only the *part* $\mathbf{P}\mathbf{x}$ *is reliable*. It is this associative mapping that we will exploit in applications of the PSOM.

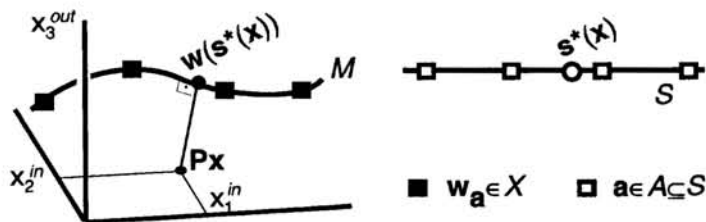

$M$ is constructed as a manifold that passes through a given set $D$ of data examples (Fig. 1 depicts the situation schematically). To this end, we assign to each data sample a point $\mathbf{a} \in S$ and denote the associated data sample by $\mathbf{w_a}$. The set $\mathbf{A}$ of the assigned parameter values $\mathbf{a}$ should provide a good discrete "model" of the topology of our data set (Fig. 1 right). The assignment between data vectors and points $\mathbf{a}$ must be made

Figure 1: Best-match $\mathbf{s}^*$ and associative completion $\mathbf{w}(\mathbf{s}^*(\mathbf{x}))$ of input $x_1, x_2$ ($\mathbf{P}\mathbf{x}$) given in the input subspace $X^{in}$. Here in this simple case, the $m = 1$ dimensional manifold $M$ is constructed to pass through four data vectors (square marked). The *left* side shows the $d = 3$ dimensional embedding space $X = X^{in} \times X^{out}$ and the *right* side depicts the best match parameter $\mathbf{s}^*(\mathbf{x})$ parameter manifold $S$ together with the "hyper-lattice" $\mathbf{A}$ of parameter values (indicated by white squares) belonging to the data vectors.

in a topology preserving fashion to ensure good interpolation by the manifold $M$ that is obtained by the following steps.

For each point $\mathbf{a} \in \mathbf{A}$, we construct a "basis function" $H(\cdot, \mathbf{a}; \mathbf{A})$ or simplified[1] $H(\cdot, \mathbf{a})$ : $S \mapsto \mathbb{R}$ that obeys *(i)* $H(\mathbf{a}_i, \mathbf{a}_j) = 1$ for $i = j$ and vanishes at all other points of $\mathbf{A}$ $i \neq j$ (orthonormality condition,) and *(ii)* $\sum_{\mathbf{a} \in \mathbf{A}} H(\mathbf{a}, \mathbf{s}) = 1$ for $\forall \mathbf{s}$ ("partition of unity" condition.) We will mainly be concerned with the case of $\mathbf{A}$ being a $m$-dimensional rectangular hyper-lattice; in this case, the functions $H(\cdot, \mathbf{a})$ can be constructed as products of Lagrange interpolation polynomials, see [11]. Then,

$$\mathbf{w}(\mathbf{s}) = \sum_{\mathbf{a} \in \mathbf{A}} H(\mathbf{s}, \mathbf{a}) \, \mathbf{w_a}. \tag{2}$$

defines a manifold $M$ that passes through all data examples. Minimizing $dist(\cdot)$ in Eq. 1 can be done by some iterative procedure, such as gradient descent or – preferably – the Levenberg-Marquardt algorithm [11]. This makes $M$ into the attractor manifold of a (discrete time) dynamical system. Since $M$ contains the data set $D$, any at least $m$-dimensional "fragment" of a data example $\mathbf{x} = \mathbf{w} \in D$ will be attracted to the correct completion $\mathbf{w}$. Inputs $\mathbf{x} \notin D$ will be attracted to some approximating manifold point.

This approach is in many ways the continuous analog of the standard discrete self-organizing map. Particularly attractive features are *(i)* that the construction of the map manifold is direct from a *small set* of training vectors, without any need for time consuming adaptation sequences, *(ii)* the capability of associative completion, which allows to freely redefine variables as inputs or outputs (by changing $dist(\cdot)$ on demand, e.g. one can reverse the mapping direction), and *(iii)* the possibility of having *attractor manifolds* instead of just attractor points.

## 3  Hierarchical PSOMs: Structuring Learning

Rapid learning requires that the structure of the learner is well matched to his task. However, if one does not want to pre-structure the learner by hand, learning again seems to be the only way to achieve the necessary pre-structuring. This leads to the idea of *structuring learning itself* and motivates to *split* learning into two stages:
*(i)* The earlier stage is considered as an *"investment stage"* that may be slow and that may require a larger number of examples. It has the task to pre-structure the system in such a way that in the later stage,
*(ii)* the now specialized system can learn *fast* and with extremely few examples.

To be concrete, we consider specialized mappings or "skills", which are dependent on the state of the system or system environment. Pre-structuring the system is achieved by learning a set of basis mappings, each in a prototypical system context or environment state ("investment phase".) This imposes a strong need for an efficient learning tool – efficient in particular with respect to the number of required training data points.

The PSOM networks appears as a very attractive solution: Fig. 2 shows a hierarchical arrangement of two PSOM. The task of mapping from input to output spaces is learned – and performed, by the "Transformation-PSOM" ("T-PSOM").

During the first learning stage, the *investment learning* phase the T-PSOM is used to learn a set of basis mappings $T_j : \vec{x}_1 \leftrightarrow \vec{x}_2$ or context dependent "skills" is constructed in the "T-PSOM", each of which gets encoded as a internal parameter or "weight" set $\omega_j$. The

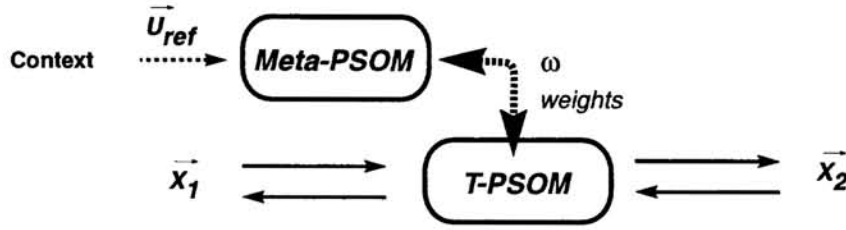

Figure 2: The transforming "T-PSOM" maps between input and output spaces (changing direction on demand). In a particular environmental context, the correct transformation is learned, and encoded in the internal parameter or weight set $\omega$. Together with an characteristic environment observation $\vec{u}_{\text{ref}}$, the weight set $\omega$ is employed as a training vector for the second level "Meta-PSOM". After learning a structured set of mappings, the Meta-PSOM is able to generalizing the mapping for a new environment. When encountering any change, the environment observation $\vec{u}_{\text{ref}}$ gives input to the Meta-PSOM and determines the new weight set $\omega$ for the basis T-PSOM.

second level PSOM ("Meta-PSOM") is responsible for learning the association between the weight sets $\omega_j$ of the first level T-PSOM and their situational contexts.

The system context is characterized by a suitable environment observation, denoted $\vec{u}_{ref}$, see Fig. 2.

The context situations are chosen such that the associated basis mappings capture already a significant amount of the underlying model structure, while still being sufficiently general to capture the variations with respect to which system environment identification is desired. For the training of the second level Meta-PSOM each constructed T-PSOM weight set $\omega_j$ serves together with its associated environment observation $\vec{u}_{ref,j}$ as a high dimensional training data vector.

*Rapid learning* is the return on invested effort in the longer pre-training phase. As a result, the task of learning the "skill" associated with an unknown system context now takes the form of an *immediate* Meta-PSOM $\rightarrow$ T-PSOM mapping: the Meta-PSOM maps the new system context observation $\vec{u}_{ref,new}$ into the parameter set $\omega_{new}$ for the T-PSOM. Equipped with $\omega_{new}$, the T-PSOM provides the desired mapping $T_{new}$.

## 4  Rapid Learning of a Stereo Visuo-motor Map

In the following, we demonstrate the potential of the investment learning approach with the task of fast learning of 3D visuo-motor maps for a robot manipulator seen by a pair of movable cameras. Thus, in this demonstration, each situated context is given by a particular camera arrangement, and the assicuated "skill" is the mapping between camera and robot coordinates.

The Puma robot is positioned behind a table and the entire scene is displayed on two windows on a computer monitor. By mouse-pointing, a user can, for example, select on the monitor one point and the position on a line appearing in the other window, to indicate a good position for the robot end effector, see Fig. 3. This requires to compute the transformation $T$ between pixel coordinates $\vec{u} = (\vec{u}^L, \vec{u}^R)$ on the monitor images and corresponding world coordinates $\vec{x}$ in the robot reference frame – or alternatively – the corresponding six robot joint angles $\vec{\theta}$ (6 DOF). Here we demonstrate an integrated solution, offering both solutions with the same network.

The T-PSOM learns each individual basis mapping $T_j$ by visiting a rectangular grid set of end effector positions $\xi_i$ (here a 3×3×3 grid in $\vec{x}$ of size $40 \times 40 \times 30 \text{cm}^3$) jointly with

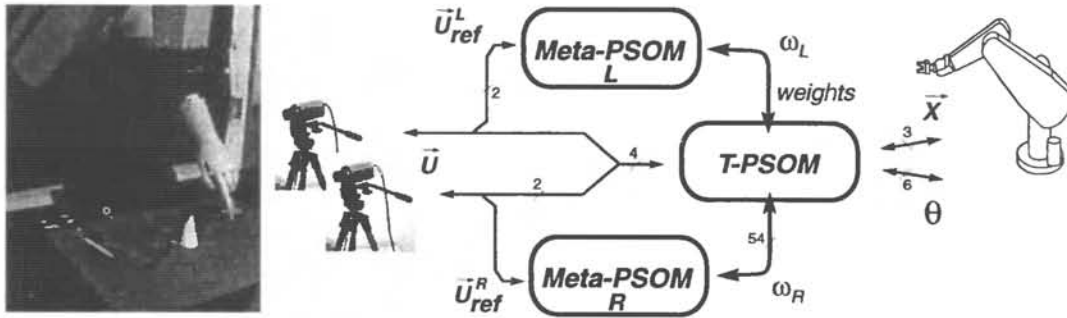

Figure 3: Rapid learning of the 3D visuo-motor coordination for two cameras. The basis T-PSOM ($m = 3$) is capable of mapping to and from three coordinate systems: Cartesian robot world co-ordinates, the robot joint angles (6-DOF), and the location of the end-effector in coordinates of the two camera retinas. Since the left and right camera can be relocated independently, the weight set of T-PSOM is split, and parts $\omega_L, \omega_R$ are learned in two separate Meta-PSOMs ("L" and "R").

the joint angle tuple $\vec{\theta}_j$ and the location in camera retina coordinates (2D in each camera) $\vec{u}_j^L, \vec{u}_j^R$. Thus the training vectors $\mathbf{w_{a_i}}$ for the construction of the T-PSOM are the tuples $(\vec{x}_i, \vec{\theta}_i, \vec{u}_i^L, \vec{u}_i^R)$.

However, each $T_j$ solves the mapping task only for the current camera arrangement, for which $T_j$ was learned. Thus there is not yet any particular advantage to other, specialized methods for camera calibration [1]. The important point is, that we now will employ the Meta-PSOM to *interpolate* in the space of the mappings $\{T_j\}$.

To keep the number of prototype mappings manageable, we reduce some DOFs of the cameras by calling for fixed focal length, camera tripod height, and twist joint. To constrain the elevation and azimuth viewing angle, we require one land mark $\xi_{fix}$ to remain visible in a constant image position. This leaves two free parameters per camera, that can now be determined by *one* extra observation of a chosen auxiliary world reference point $\xi_{ref}$. We denote the camera image coordinates of $\xi_{ref}$ by $\vec{u}_{ref} = (\vec{u}_{ref}^L, \vec{u}_{ref}^R)$. By reuse of the cameras as "environment sensor", $\vec{u}_{ref}$ now implicitly encodes the two camera positions.

In the investing pre-training phase, nine mappings $T_j$ are learned by the T-PSOM, each camera visiting a $3 \times 3$ grid, sharing the set of visited robot positions $\xi_i$. As Fig. 2 suggests, normally the entire weight set $\omega$ serves as part of the training vector to the Meta-PSOM. Here the problem becomes factorized since the left and right camera change tripod place independently: the weight set of the T-PSOM is split, and the two parts can be learned in separate Meta-PSOMs. Each training vector $\mathbf{w_{a_j}}$ for the left camera Meta-PSOM consists of the context observation $\vec{u}_{ref}^L$ and the T-PSOM weight set part $\omega_L = (\vec{u}_1^L, \ldots, \vec{u}_{27}^L)$ (analogous the right camera Meta-PSOM.)

This enables in the following phase the *rapid learning*, for new, unknown camera places. On the basis of *one single* observation $\vec{u}_{ref}$, the desired transformation $T$ is constructed. As visualized in Fig. 3, $\vec{u}_{ref}$ serves as the input to the second level Meta-PSOMs. Their outputs are interpolations between previously learned weight sets and they project directly into the weight set of the basis level T-PSOM.

The resulting T-PSOM can map in various directions. This is achieved by specifying a suitable distance function $dist(\cdot)$ via the projection matrix $\mathbf{P}$, e.g.:

$$\vec{x}(\vec{u}) \quad = \quad F_{T-PSOM}^{u \mapsto x}(\vec{u}; \quad \omega_L(\vec{u}_{ref}^L), \omega_R(\vec{u}_{ref}^R)) \tag{3}$$

$$\vec{\theta}(\vec{u}) = F^{u\mapsto\theta}_{T-PSOM}(\vec{u}; \quad \omega_L(\vec{u}^L_{ref}), \omega_R(\vec{u}^R_{ref})) \tag{4}$$

$$\vec{u}(\vec{x}) = F^{x\mapsto u}_{T-PSOM}(\vec{x}; \quad \omega_L(\vec{u}^L_{ref}), \omega_R(\vec{u}^R_{ref})) \tag{5}$$

$$\omega_L(\vec{u}^L_{ref}) = F^{u\mapsto\omega}_{Meta-PSOM,L}(\vec{u}^L_{ref}; \quad \Omega_L); \quad \text{analog } \omega_R(\vec{u}^R_{ref}) \tag{6}$$

Table 1 shows experimental results averaged over 100 random locations $\xi$ (from within the range of the training set) seen in 10 different camera set-ups, from within the $3\times3$ square grid of the training positions, located in a normal distance of about 125 cm (center to work space center, $1\,m^2$, total range of about 55–210 cm), covering a disparity angle range of $25°$–$150°$. For identification of the positions $\xi$ in image coordinates, a tiny light source was installed at the manipulator tip and a simple procedure automated the finding of $\vec{u}$ with about $\pm0.8$ pixel accuracy. For the achieved precision it is important to share the same set of robot positions $\xi_i$, and that the sets are topologically ordered, here as a 3×3×3 goal position grid ($i$) and two $3\times3$ camera location ($j$) grids.

| Mapping Direction | Direct trained T-PSOM | | T-PSOM with Meta-PSOM | |
|---|---|---|---|---|
| pixel $\vec{u}\mapsto\vec{x}_{robot}$ $\Rightarrow$ Cartesian error $\Delta\vec{x}$ | 1.4 mm | 0.008 | 4.4 mm | 0.025 |
| Cartesian $\vec{x}\mapsto\vec{u}$ $\Rightarrow$ pixel error | 1.2 pix | 0.010 | 3.3 pix | 0.025 |
| pixel $\vec{u}\mapsto\vec{\theta}_{robot}$ $\Rightarrow$ Cartesian error $\Delta\vec{x}$ | 3.8 mm | 0.023 | 5.4 mm | 0.030 |

Table 1: Mean Euclidean deviation (mm or pixel) and normalized root mean square error (NRMS) for 1000 points total in comparison of a direct trained T-PSOM and the described hierarchical Meta-PSOM network, in the rapid learning mode after *one single* observation.

## 5   Discussion and Conclusion

A crucial question is how to structure systems, such that learning can be efficient. In the present paper, we demonstrated a hierarchical approach that is motivated by a *decomposition of the learning phase into two different stages:* A longer, initial learning phase "invests" effort into a gradual and domain-specific specialization of the system. This *investment learning* does not yet produce the final solution, but instead pre-structures the system such that the subsequently final specialization to a particular solution (within the chosen domain) can be achieved extremely rapidly.

To implement this approach, we used a hierarchical architecture of mappings. While in principle various kinds of network types could be used for this mappings, a practically feasible solution must be based on a network type that allows to construct the required basis mappings from rather small number of training examples. In addition, since we use interpolation in weight space, similar mappings should give rise to similar weight sets to make interpolation meaningful. PSOM meat this requirements very well, since they allow a direct non-iterative construction of smooth mappings from rather small data sets. They achieve this be generalizing the discrete self-organizing map [3, 9] into a continuous map manifold such that interpolation for new data points can benefit from topology information that is not available to most other methods.

While PSOMs resemble local models [4, 5, 6] in that there is no interference between different training points, their use of a orthogonal set of basis functions to construct the

map manifold put them in a intermediate position between the extremes of local and of fully distributed models.

A further very useful property in the present context is the ability of PSOMs to work as an attractor network with a continuous attractor manifold. Thus a PSOM needs no fixed designation of variables as inputs and outputs; Instead the projection matrix $\mathbf{P}$ can be used to freely partition the full set of variables into input and output values. Values of the latter are obtained by a process of associative completion.

Technically, the investment learning phase is realized by learning a set of *prototypical basis mappings* represented as weight sets of a T-PSOM that attempt to cover the range of tasks in the given domain. The capability for subsequent rapid specialization within the domain is then provided by an additional mapping that maps a situational context into a suitable combination of the previously learned prototypical basis mappings. The construction of this mapping again is solved with a PSOM ("Meta"-PSOM) that *interpolates in the space of prototypical basis mappings* that were constructed during the "investment phase".

We demonstrated the potential of this approach with the task of 3D visuo-motor mapping, learn-able with a single observation after repositioning a pair of cameras.
The achieved accuracy of 4.4 mm after learning by a single observation, compares very well with the distance range 0.5–2.1 m of traversed positions. As further data becomes available, the T-PSOM can certainly be fine-tuned to improve the performance to the level of the directly trained T-PSOM.

The presented arrangement of a basis T-PSOM and two Meta-PSOMs demonstrates further the possibility to split hierarchical learning in independently changing domain sets. When the number of involved free context parameters is growing, this factorization is increasingly crucial to keep the number of pre-trained prototype mappings manageable.

## Footnotes

[1]In contrast to kernel methods, the basis functions may depend on the relative position to all other knots. However, we drop in our notation the dependency $H(\mathbf{a}, \mathbf{s}) = H(\mathbf{a}, \mathbf{s}; \mathbf{A})$ on the latter.

# References

[1] K. Fu, R. Gonzalez and C. Lee. *Robotics : Control, Sensing, Vision, and Intelligence.* McGraw-Hill, 1987

[2] F. Girosi and T. Poggio. Networks and the best approximation property. *Biol. Cybern.,* 63(3):169–176, 1990.

[3] T. Kohonen. *Self-Organization and Associative Memory.* Springer, Heidelberg, 1984.

[4] J. Moody and C. Darken. Fast learning in networks of locally-tuned processing units. *Neural Computation,* 1:281–294, 1989.

[5] S. Omohundro. Bumptrees for efficient function, constraint, and classification learning. In *NIPS*3,* pages 693–699. Morgan Kaufman Publishers, 1991.

[6] J. Platt. A resource-allocating network for function interpolation. *Neural Computation,* 3:213–255, 1991

[7] M. Powell. *Radial basis functions for multivariable interpolation: A review,* pages 143–167. Clarendon Press, Oxford, 1987.

[8] H. Ritter. Parametrized self-organizing maps. In S. Gielen and B. Kappen, editors, *ICANN'93-Proceedings, Amsterdam,* pages 568–575. Springer Verlag, Berlin, 1993.

[9] H. Ritter, T. Martinetz, and K. Schulten. *Neural Computation and Self-organizing Maps.* Addison Wesley, 1992.

[10] J. Walter and H. Ritter. Local PSOMs and Chebyshev PSOMs – improving the parametrised self-organizing maps. In *Proc. ICANN, Paris,* volume 1, pages 95–102, October 1995.

[11] J. Walter and H. Ritter. Rapid learning with parametrized self-organizing maps. *Neurocomputing, Special Issue,* (in press), 1996.
